# Asymptotic slowing down of the nearest-neighbor classifier

**Robert R. Snapp**
CS/EE Department
University of Vermont
Burlington, VT 05405

**Demetri Psaltis**
Electrical Engineering
Caltech 116–81
Pasadena, CA 91125

**Santosh S. Venkatesh**
Electrical Engineering
University of Pennsylvania
Philadelphia, PA 19104

## Abstract

If patterns are drawn from an $n$-dimensional feature space according to a probability distribution that obeys a weak smoothness criterion, we show that the probability that a random input pattern is misclassified by a nearest-neighbor classifier using $M$ random reference patterns asymptotically satisfies

$$P_M(\text{error}) \sim P_\infty(\text{error}) + \frac{a}{M^{2/n}},$$

for sufficiently large values of $M$. Here, $P_\infty(\text{error})$ denotes the probability of error in the infinite sample limit, and is at most twice the error of a Bayes classifier. Although the value of the coefficient $a$ depends upon the underlying probability distributions, the exponent of $M$ is largely distribution free. We thus obtain a concise relation between a classifier's ability to generalize from a finite reference sample and the dimensionality of the feature space, as well as an analytic validation of Bellman's well known "curse of dimensionality."

## 1  INTRODUCTION

One of the primary tasks assigned to neural networks is pattern classification. Common applications include recognition problems dealing with speech, handwritten characters, DNA sequences, military targets, and (in this conference) sexual identity. Two fundamental concepts associated with pattern classification are *generalization* (how well does a classifier respond to input data it has never encountered before?) and *scalability* (how are a classifier's processing and training requirements affected by increasing the number of features that describe the input patterns?).

Despite recent progress, our present understanding of these concepts in the context of neural networks is obstructed by complexities in the functional form of the network and in the classification problems themselves.

In this correspondence we will present analytic results on these issues for the nearest-neighbor classifier. Noted for its algorithmic simplicity and nearly optimal performance in the infinite sample limit, this pattern classifier plays a central role in the field of pattern recognition. Furthermore, because it uses proximity in feature space as a measure of class similarity, its performance on a given classification problem should yield qualitative cues to the performance of a neural network. Indeed, a nearest-neighbor classifier can be readily implemented as a "winner-take-all" neural network.

## 2   THE TASK OF PATTERN CLASSIFICATION

We begin with a formulation of the two-class problem (Duda and Hart, 1973):

> Let the labels $\omega_1$ and $\omega_2$ denote two states of nature, or pattern classes. A pattern belonging to one of these two classes is selected, and a vector of $n$ features, $\mathbf{x}$, that describe the selected pattern is presented to a pattern classifier. The classifier then attempts to guess the selected pattern's class by assigning $\mathbf{x}$ to either $\omega_1$ or $\omega_2$.

As an example, the two class labels might represent the states *benign* and *malignant* as they pertain to the diagnosis of cancer tumors; the feature vector could then be a $1024 \times 1024$ pixel, real-valued representation of an electron-microscope image. A pattern classifier can thus be viewed as a mapping from an $n$-dimensional feature space to the discrete set $\{\omega_1, \omega_2\}$, and can be specified by demarcating the regions in the $n$-dimensional feature space that correspond to $\omega_1$ and $\omega_2$. We define the decision region $\mathcal{R}_1$ as the set of feature vectors that the pattern classifier assigns to $\omega_1$, with an analogous definition for $\mathcal{R}_2$. A useful figure of merit is the probability that the feature vector of a randomly selected pattern is assigned to the correct class.

### 2.1   THE BAYES CLASSIFIER

If sufficient information is available, it is possible to construct an optimal pattern classifier. Let $P(\omega_1)$ and $P(\omega_2)$ denote the *prior probabilities* of the two states of nature. (For our cancer diagnosis problem, the prior probabilities can be estimated by the relative frequency of each type of tumor in a large statistical sample.) Further, let $p(\mathbf{x} \mid \omega_1)$ and $p(\mathbf{x} \mid \omega_2)$ denote the *class-conditional probability densities* of the feature vector for the two class problem. The total probability density is now defined by $p(\mathbf{x}) = p(\mathbf{x} \mid \omega_1)P(\omega_1) + p(\mathbf{x} \mid \omega_2)P(\omega_2)$, and gives the unconditional distribution of the feature vector. Where $p(\mathbf{x}) \neq 0$ we can now use Bayes' rule to compute the *posterior probabilities*:

$$P(\omega_1 \mid \mathbf{x}) = \frac{p(\mathbf{x} \mid \omega_1)P(\omega_1)}{p(\mathbf{x})} \qquad \text{and} \qquad P(\omega_2 \mid \mathbf{x}) = \frac{p(\mathbf{x} \mid \omega_2)P(\omega_2)}{p(\mathbf{x})}.$$

*The Bayes classifier* assigns an unclassified feature vector $\mathbf{x}$ to the class label having

the greatest posterior probability. (If the posterior probabilities happen to be equal, then the class assignment is arbitrary.) With $\mathcal{R}_1$ and $\mathcal{R}_2$ denoting the two decision regions induced by this strategy, the probability of error of the Bayes classifier, $P_B$, is just the probability that $\mathbf{x}$ is drawn from class $\omega_1$ but lies in the Bayes decision region $\mathcal{R}_2$, or conversely, that $\mathbf{x}$ is drawn from class $\omega_2$ but lies in the Bayes decision region $\mathcal{R}_1$:

$$P_B = \int_{\mathcal{R}_2} P(\omega_1 \mid \mathbf{x}) \, p(\mathbf{x}) \, d^n x + \int_{\mathcal{R}_1} P(\omega_2 \mid \mathbf{x}) \, p(\mathbf{x}) \, d^n x.$$

The reader may verify that the Bayes classifier minimizes the probability of error.

Unfortunately, it is usually impossible to obtain expressions for the class-conditional densities and prior probabilities in practice. Typically, the available information resides in a set of correctly labeled patterns, which we collectively term a *training* or *reference sample*. Over the last few decades, numerous pattern classification strategies have been developed that attempt to learn the structure of a classification problem from a finite training sample. (The backpropagation algorithm is a recent example.) The underlying hope is that the classifier's performance can be made acceptable with a sufficiently large reference sample. In order to understand how large a sample may be needed, we turn to what is perhaps the simplest learning algorithm of this class.

## 3    THE NEAREST-NEIGHBOR CLASSIFIER

Let $\mathcal{X}_M = \{(\mathbf{x}^{(1)}, \theta^{(1)}), (\mathbf{x}^{(2)}, \theta^{(2)}), \dots, (\mathbf{x}^{(M)}, \theta^{(M)})\}$ denote a finite reference sample of $M$ feature vectors, $\mathbf{x}^{(i)} \in \mathbf{R}^n$, with corresponding known class assignments, $\theta^{(i)} \in \{\omega_1, \omega_2\}$. The *nearest-neighbor rule* assigns each feature vector $\mathbf{x}$ to class $\omega_1$ or $\omega_2$ as a function of the reference $M$-sample as follows:

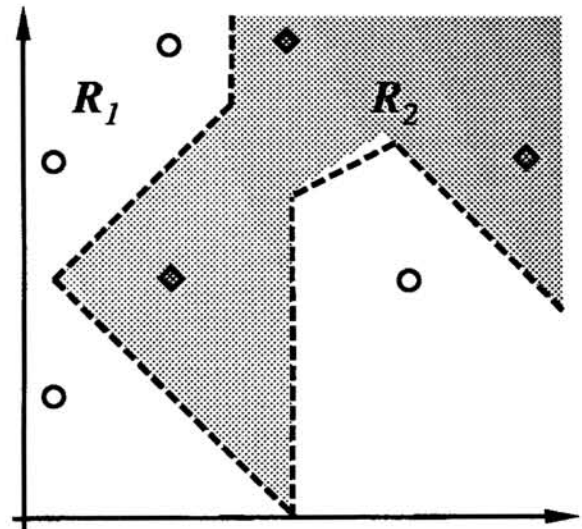

- *Identify* $(\mathbf{x}', \theta') \in \mathcal{X}_M$ such that $\|\mathbf{x} - \mathbf{x}'\| \leq \|\mathbf{x} - \mathbf{x}^{(i)}\|$ for $i$ ranging from 1 through $M$;

- *Assign* $\mathbf{x}$ to class $\theta'$.

Here, $\|\mathbf{x} - \mathbf{y}\| = \sqrt{\sum_{j=1}^{n}(x_j - y_j)^2}$ denotes the Euclidean metric in $\mathbf{R}^n$.[1] The nearest-neighbor rule hence classifies each feature vector $\mathbf{x}$ according to the label, $\theta'$, of the closest point, $\mathbf{x}'$, in the reference sample. As an example, we sketch the nearest-neighbor decision regions for a two-dimensional classification problem in Fig. 1.

Figure 1: The decision regions induced by a nearest-neighbor classifier with a seven-element reference set in the plane.

It is interesting to consider how the performance of this classifier compares with that of a Bayes classifier. To facilitate this analysis, we assume that the reference patterns are selected from the total probability density $p(\mathbf{x})$ in a statistically independent manner (i.e., the choice of $\mathcal{X}_j$ does not in any way bias the selection of $\mathbf{x}^{(j+1)}$ and $\theta^{(j+1)}$). Furthermore, let $P_M(\text{error})$ denote the probability of error of a nearest-neighbor classifier working with the reference sample $\mathcal{X}_M$, and let $P_\infty(\text{error})$ denote this probability in the infinite sample limit ($M \to \infty$). We will also let $\mathcal{S}$ denote the volume in feature space over which $p(\mathbf{x})$ is nonzero. The following well known theorem shows that the nearest-neighbor classifier, with an infinite reference sample, is nearly optimal (Cover and Hart, 1967).[2]

**Theorem 1** *For the two-class problem in the infinite sample limit, the probability of error of a nearest-neighbor classifier tends toward the value,*

$$P_\infty(\text{error}) = 2 \int_{\mathcal{S}} P(\omega_1 \mid \mathbf{x}) P(\omega_2 \mid \mathbf{x}) p(\mathbf{x}) \, d^n x,$$

*which is furthermore bounded by the two inequalities,*

$$P_B \le P_\infty(\text{error}) \le 2P_B(1 - P_B),$$

*where $P_B$ is the probability of error of a Bayes classifier.*

This encouraging result is not so surprising if one considers that, with probability one, about every feature vector $\mathbf{x}$ is centered a ball of radius $\epsilon$ that contains an *infinite* number of reference feature vectors for every $\epsilon > 0$. The annoying factor of two accounts for the event that the nearest neighbor to $\mathbf{x}$ belongs to the class with smaller posterior probability.

## 3.1    THE ASYMPTOTIC CONVERGENCE RATE

In order to satisfactorily address the issues of generalization and scalability for the nearest-neighbor classifier, we need to consider the rate at which the performance of the classifier approaches its infinite sample limit. The following theorem applicable to nearest-neighbor classification in *one-dimensional* feature spaces was shown by Cover (1968).

**Theorem 2** *Let $p(x \mid \omega_1)$ and $p(x \mid \omega_2)$ have uniformly bounded third derivatives and let $p(\mathbf{x})$ be bounded away from zero on $\mathcal{S}$. Then for sufficiently large $M$,*

$$P_M(\text{error}) = P_\infty(\text{error}) + O\left(\frac{1}{M^2}\right).$$

Note that this result is also very encouraging in that an order of magnitude increase in the sample size, decreases the error rate by *two* orders of magnitude.

The following theorem is our main result which extends Cover's theorem to $n$-dimensional feature spaces:

**Theorem 3** *Let $p(\mathbf{x} \mid \omega_1)$, $p(\mathbf{x} \mid \omega_2)$, and $p(\mathbf{x})$ satisfy the same conditions as in Theorem 2. Then, there exists a scalar a (depending on n) such that*

$$P_M(\text{error}) \sim P_\infty(\text{error}) + \frac{a}{M^{2/n}},$$

*where the right-hand side describes the first two terms of an asymptotic expansion in reciprocal powers of $M^{2/n}$. Explicitly,*

$$a = \frac{\Gamma\left(1 + \frac{2}{n}\right)\left(\Gamma\left(\frac{n}{2} + 1\right)\right)^{2/n}}{n\pi} \sum_{i=1}^{n} \int_S \left(\frac{\beta_i(\mathbf{x})p_i(\mathbf{x})}{p(\mathbf{x})} + \frac{1}{2}\gamma_{ii}(\mathbf{x})\right)(p(\mathbf{x}))^{1-2/n}\, d^n\mathbf{x}.$$

*where,*

$$p_i(\mathbf{x}) = \frac{\partial p(\mathbf{x})}{\partial x_i}$$

$$\beta_i(\mathbf{x}) = P(\omega_1 \mid \mathbf{x})\frac{\partial P(\omega_2 \mid \mathbf{x})}{\partial x_i} + \frac{\partial P(\omega_1 \mid \mathbf{x})}{\partial x_i}P(\omega_2 \mid \mathbf{x})$$

$$\gamma_{ii}(\mathbf{x}) = P(\omega_1 \mid \mathbf{x})\frac{\partial^2 P(\omega_2 \mid \mathbf{x})}{\partial x_i^2} + \frac{\partial^2 P(\omega_1 \mid \mathbf{x})}{\partial x_i^2}P(\omega_2 \mid \mathbf{x}).$$

For $n = 1$ this result agrees with Cover's theorem. With increasing $n$, however, the convergence rate significantly slows down. Note that the constant $a$ depends on the way in which the class-conditional densities overlap. If $a$ is bounded away from zero, then for sufficiently small $\delta > 0$, $P_M(\text{error}) - P_\infty(\text{error}) < \delta$ is satisfied only if $M > (a/\delta)^{n/2}$ so that the sample size required to achieve a given performance criterion is exponential in the dimensionality of the feature space. The above provides a sufficient condition for Bellman's well known "curse of dimensionality" in this context.

It is also interesting to note that one can easily construct classification problems for which $a$ vanishes. (Consider, for example, $p(\mathbf{x} \mid \omega_1) = p(\mathbf{x} \mid \omega_2)$ for all $\mathbf{x}$.) In these cases the higher-order terms in the asymptotic expansion are important.

## 4    A NUMERICAL EXPERIMENT

A conspicuous weakness in the above theorem is the requirement that $p(\mathbf{x})$ be bounded away from zero over $S$. In exchange for a uniformly convergent asymptotic expansion, we have omitted many important probability distributions, including normal distributions. Therefore we numerically estimate the asymptotic behavior of $P_M(\text{error})$ for a problem consisting of two normally distributed classes in $\mathbf{R}^n$:

$$p(\mathbf{x} \mid \omega_1) = \frac{1}{(2\pi\sigma^2)^{n/2}}\exp\left[-\frac{1}{2\sigma^2}\left((x_1 - \mu)^2 + \sum_{j=2}^{n} x_j^2\right)\right],$$

$$p(\mathbf{x} \mid \omega_2) = \frac{1}{(2\pi\sigma^2)^{n/2}}\exp\left[-\frac{1}{2\sigma^2}\left((x_1 + \mu)^2 + \sum_{j=2}^{n} x_j^2\right)\right].$$

Assuming that $P(\omega_1) = P(\omega_2) = 1/2$, we find

$$P_\infty(\text{error}) = \frac{1}{\sigma\sqrt{2\pi}}e^{-\mu^2/2\sigma^2}\int_0^\infty e^{-x^2/2\sigma^2}\operatorname{sech}\left(\frac{\mu x}{\sigma^2}\right)dx.$$

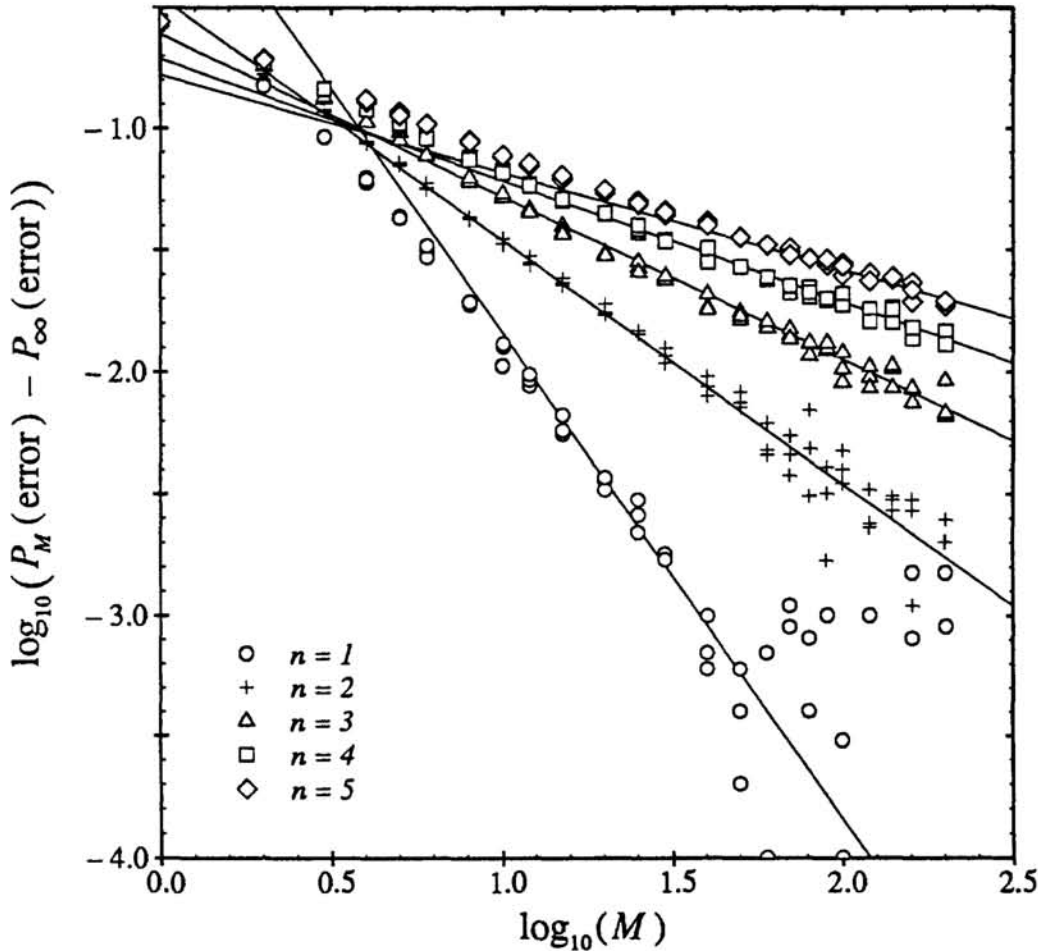

Figure 2: Numerical validation of the nearest-neighbor scaling hypothesis for two normally distributed classes in $\mathbf{R}^n$.

For $\mu = \sigma = 1$, $P_\infty(\text{error})$ is numerically found to be 0.22480, which is consistent with the Bayes probability of error, $P_B = (1/2)\text{erfc}(1/\sqrt{2}) = 0.15865$. (Note that the expression for $a$ given in Theorem 3 is undefined for these distributions.) For $n$ ranging from 1 to 5, and $M$ ranging from 1 to 200, three estimates of $P_M(\text{error})$ were obtained, each as the fraction of "failures" in 160,000 or more Bernoulli trials. Each trial consists of constructing a pseudo-random sample of $M$ reference patterns, followed by a single attempt to correctly classify a random input pattern. These estimates of $P_M$ are represented in Figure 2 by circular markers for $n = 1$, crosses for $n = 2$, etc. The lines in Figure 2 depict the power law

$$P_M(\text{error}) = P_\infty(\text{error}) + bM^{-2/n},$$

where, for each $n$, $b$ is chosen to obtain an appealing fit. The agreement between these lines and data points suggests that the asymptotic scaling hypothesis of Theorem 3 can be extended to a wider class of distributions.

## 5   DISCUSSION

The preceding analysis indicates that the convergence rate of the nearest-neighbor classifier slows down dramatically as the dimensionality of the feature space increases. This rate reduction suggests that proximity in feature space is a less effective measure of class identity in higher dimensional feature spaces. It is also clear that some degree of smoothness in the class-conditional densities is necessary, as well as sufficient, for the asymptotic behavior described by our analysis to occur: in the absence of smoothness conditions, one can construct classification problems for which the nearest-neighbor convergence rate is arbitrarily slow, even in one dimension (Cover, 1968). Fortunately, the most pressing classification problems are typically smooth in that they are constrained by regularities implicit in the laws of nature (Marr, 1982). With additional prior information, the convergence rate may be enhanced by selecting a fewer number of descriptive features.

Because of their smooth input-output response, neural networks appear to use proximity in feature space as a basis for classification. One might, therefore, expect the required sample size to scale exponentially with the dimensionality of the feature space. Recent results from computational learning theory, however, imply that with a sample size proportional to the *capacity*—a combinatorial quantity which is characteristic of the network architecture and which typically grows polynomially in the dimensionality of the feature space—one can in principle identify network parameters (weights) which give (close to) the smallest classification error for the given architecture (Baum and Haussler, 1989). There are two caveats, however. First, the information-theoretic sample complexities predicted by learning theory give no clue as to whether, given a sample of the requisite size, there exist any *algorithms* that can specify the appropriate parameters in a reasonable time frame. Second, and more fundamental, one cannot in general determine whether a particular architecture is intrinsically well suited to a given classification problem. The best performance achievable may be substantially poorer than that of a Bayes classifier. Thus, without sufficient prior information, one must search through the space of all possible network architectures for one that does fit the problem well. This situation now effectively resembles a non-parametric classifier and the analytic results for the sample complexities of the nearest-neighbor classifier should provide at least qualitative indications of the corresponding case for neural networks.

## Footnotes

[1]Other metrics, such as the more general Minkowski-$r$ metric, are also possible.

[2]Originally, this theorem was stated for multiclass decision problems; it is here presented for the two class problem only for simplicity.

### References

Baum, E. B. and Haussler, D. (1989), "What size net gives valid generalization," *Neural Computation*, 1, pp. 151–160.

Cover, T. M. (1968), "Rates of convergence of nearest neighbor decision procedures," *Proc. First Annual Hawaii Conference on Systems Theory*, pp. 413–415.

Cover, T. M. and P. E. Hart (1967), "Nearest neighbor pattern classification," *IEEE Trans. Info. Theory*, vol. IT–13, pp. 21–27.

Duda, R. O. and P. E. Hart (1973), *Pattern Classification and Scene Analysis*. New York: John Wiley & Sons.

Marr, D. (1982), *Vision*, San Francisco: W. H. Freeman.
